# EM-DD: An Improved Multiple-Instance Learning Technique

**Qi Zhang**
Department of Computer Science
Washington University
St. Louis, MO  63130-4899
*qz@cs.wustl.edu*

**Sally A. Goldman**
Department of Computer Science
Washington University
St. Louis, MO  63130-4899
*sg@cs.wustl.edu*

## Abstract

We present a new multiple-instance (MI) learning technique (EM-DD) that combines EM with the diverse density (DD) algorithm. EM-DD is a general-purpose MI algorithm that can be applied with boolean or real-value labels and makes real-value predictions. On the boolean Musk benchmarks, the EM-DD algorithm without any tuning significantly outperforms all previous algorithms. EM-DD is relatively insensitive to the number of relevant attributes in the data set and scales up well to large bag sizes. Furthermore, EM-DD provides a new framework for MI learning, in which the MI problem is converted to a single-instance setting by using EM to estimate the instance responsible for the label of the bag.

## 1   Introduction

The *multiple-instance* (MI) learning model has received much attention. In this model, each training example is a set (or *bag*) of instances along with a single label equal to the maximum label among all instances in the bag. The individual instances within the bag are not given labels. The goal is to learn to accurately predict the label of previously unseen bags. Standard supervised learning can be viewed as a special case of MI learning where each bag holds a single instance. The MI learning model was originally motivated by the *drug activity prediction problem* where each instance is a possible conformation (or shape) of a molecule and each bag contains all likely low-energy conformations for the molecule. A molecule is active if it binds strongly to the target protein in at least one of its conformations and is inactive if no conformation binds to the protein. The problem is to predict the label (active or inactive) of molecules based on their conformations.

The MI learning model was first formalized by Dietterich et al. in their seminal paper [4] in which they developed MI algorithms for learning axis-parallel rectangles (APRs) and they also provided two benchmark "Musk" data sets. Following this work, there has been a significant amount of research directed towards the development of MI algorithms using different learning models [2,5,6,9,12]. Maron and

Raton [7] applied the multiple-instance model to the task of recognizing a person from a series of images that are labeled positive if they contain the person and negative otherwise. The same technique was used to learn descriptions of natural scene images (such as a waterfall) and to retrieve similar images from a large image database using the learned concept [7]. More recently, Ruffo [11] has used this model for data mining applications.

While the musk data sets have boolean labels, algorithms that can handle real-value labels are often desirable in real-world applications. For example, the binding affinity between a molecule and receptor is quantitative, and hence a real-value classification of binding strength is preferable to a binary one. Most prior research on MI learning is restricted to concept learning (i.e. boolean labels). Recently, MI learning with real-value labels has been performed using extensions of the diverse density (DD) and $k$-NN algorithms [1] and using MI regression [10].

In this paper, we present a general-purpose MI learning technique (EM-DD) that combines EM [3] with the extended DD [1] algorithm. The algorithm is applied to both boolean and real-value labeled data and the results are compared with corresponding MI learning algorithms from previous work. In addition, the effects of the number of instances per bag and the number of relevant features on the performance of EM-DD algorithm are also evaluated using artificial data sets. A second contribution of this work is a new general framework for MI learning of converting the MI problem to a single-instance setting using EM. A very similar approach was also used by Ray and Page [10].

## 2   Background

Dietterich et al. [4], presented three algorithms for learning APRs in the MI model. Their best performing algorithm (*iterated-discrim*), starts with a point in the feature space and "grows" a box with the goal of finding the smallest box that covers at least one instance from each positive bag and no instances from any negative bag. The resulting box was then expanded (via a statistical technique) to get better results. However, the *test* data from Musk1 was used to tune the parameters of the algorithm. These parameters are then used for Musk1 and Musk2.

Auer [2] presented an algorithm, MULTINST, that learns using simple statistics to find the halfspaces defining the boundaries of the target APR and hence avoids some potentially hard computational problems that were required by the heuristics used in the iterated-discrim algorithm. More recently, Wang and Zucker [11] proposed a lazy learning approach by applying two variant of the $k$ nearest neighbor algorithm ($k$-NN) which they refer to as citation-$k$NN and Bayesian $k$-NN. Ramon and De Raedt [9] developed a MI neural network algorithm.

Our work builds heavily upon the *Diverse Density* (DD) algorithm of Maron and Lozano-Pérez [5,6]. When describing the shape of a molecule by $n$ features, one can view each conformation of the molecule as a point in a $n$-dimensional feature space. The diverse density at a point $p$ in the feature space is a probabilistic measure of both how many *different* positive bags have an instance near $p$, and how far the negative instances are from $p$. Intuitively, the diversity density of a hypothesis $h$ is just the likelihood (with respect to the data) that $h$ is the target. A high diverse density indicates a good candidate for a "true" concept.

We now formally define the general MI problem (with boolean or real-value la-

bels) and DD likelihood measurement originally defined in [6] and extended to real-value labels in [1]. Let $D$ be the labeled data which consists of a set of $m$ bags $B = \{B_1, \ldots, B_m\}$ and labels $L = \{\ell_1, \ldots, \ell_m\}$, i.e., $D = \{< B_1, \ell_1 >, \ldots, < B_m, \ell_m >\}$. Let bag $B_i = \{B_{i1}, \ldots, B_{ij}, \ldots B_{in}\}$ where $B_{ij}$ denote the $j^{th}$ instance in bag $i$. Assume the labels of the instances in $B_i$ are $\ell_{i1}, \ldots, \ell_{ij}, \ldots, \ell_{in}$. For boolean labels, $\ell_i = \ell_{i1} \vee \ell_{i2} \vee \ldots \vee \ell_{in}$, and for real-value labels, $\ell_i = max\{\ell_{i1}, \ell_{i2}, \ldots, \ell_{in}\}$. The diverse density of hypothesized target point $h$ is defined as $DD(h) = \Pr(h \mid D) = \dfrac{\Pr(D \mid h)\Pr(h)}{\Pr(D)} = \dfrac{\Pr(B, L \mid h)\Pr(h)}{\Pr(B, L)}$. Assuming a uniform prior on the hypothesis space and independence of $< B_i, \ell_i >$ pairs given $h$, using Bayes' rule, the maximum likelihood hypothesis, $h_{DD}$, is defined as:

$$\arg\max_{h \in H} \Pr(D \mid h) = \arg\max_{h \in H} \prod_{i=1}^{n} \Pr(B_i, \ell_i \mid h) = \arg\min_{h \in H} \sum_{i=1}^{n} (-\log \Pr(\ell_i \mid h, B_i))$$

where $Label(B_i \mid h)$ is the label that would be given to $B_i$ if $h$ were the correct hypothesis. As in the extended DD algorithm [1], $\Pr(\ell_i \mid h, B_i)$ is estimated as $1 - |\ell_i - Label(B_i \mid h)|$ in [1]. When the labels are boolean (0 or 1), this formulation is exactly the most-likely-cause estimator used in the original DD algorithm [5]. For most applications the influence each feature has on the label varies greatly. This variation is modeled in the DD algorithm by associating with each attribute an (unknown) scale factor. Hence the target concept really consists of two values per dimension, the ideal attribute value and the scale value. Using the assumption that binding strength drops exponentially as the similarity between the conformation to the ideal shape increases, the following generative model was introduced by Maron and Lozano-Pérez [6] for estimating the label of bag $B_i$ for hypothesis $h = \{h_1, \ldots, h_n, s_1, \ldots, s_n\}$ :

$$Label(B_i \mid h) = \max_{j}\left\{exp[-\sum_{d=1}^{n} (s_d(B_{ijd} - h_d))^2]\right\} \tag{1}$$

where $s_d$ is a scale factor indicating the importance of feature $d$, $h_d$ is the feature value for dimension $d$, and $B_{ijd}$ is the feature value of instance $B_{ij}$ on dimension $d$. Let $NLDD(h, D) = \sum_{i=1}^{n}(-\log \Pr(\ell_i \mid h, B_i))$, where NLDD denote the negative logarithm of DD. The DD algorithm [6] uses a two-step gradient descent search to find a value of $h$ that minimizes NLDD (and hence maximizes DD).

Ray and Page [10] developed multiple-instance regression algorithm which can also handle real-value labeled data. They assumed an underlying linear model for the hypothesis and applied the algorithm to some artificial data. Similar to the current work, they also used EM to select one instance from each bag so multiple regression can be applied to MI learning.

## 3   Our algorithm: EM-DD

We now describe EM-DD and compare it with the original DD algorithm. One reason why MI learning is so difficult is the ambiguity caused by not knowing which instance is the important one. The basic idea behind EM-DD is to view the knowledge of which instance corresponds to the label of the bag as a missing attribute which can be estimated using EM approach in a way similar to how EM is used in the MI regression [10]. EM-DD starts with some initial guess of a target point $h$ obtained in the standard way by trying points from positive bags, then repeatedly performs the following two steps that combines EM with DD to search for the maximum likelihood hypothesis. In the first step ($E$-step), the current

hypothesis $h$ is used to pick one instance from each bag which is most likely (given our generative model) to be the one responsible for the label given to the bag. In the second step ($M$-step), we use the two-step gradient ascent search (quasi-newton search `dfpmin` in [8]) of the standard DD algorithm to find a new $h'$ that maximizes $DD(h)$. Once this maximization step is completed, we reset the proposed target $h$ to $h'$ and return to the first step until the algorithm converges. Pseudo-code for EM-DD is given in Figure 1.

We now briefly provide intuition as to why EM-DD improves both the accuracy and computation time of the DD algorithm. Again, the basic approach of DD is to use a gradient search to find a value of $h$ that maximizes DD($h$). In every search step, the DD algorithm uses all points in each bag and hence the maximum that occurs in Equation (1) must be computed. The prior diverse density algorithms [1,5,6,7] used a *softmax* approximation for the maximum (so that it will be differentiable), which dramatically increases the computation complexity and introduces additional error based on the parameter selected in softmax. In comparison, EM-DD converts the multiple-instance data to single-instance data by removing all but one point per bag in the $E$-step, which greatly simplifies the search step since the maximum that occurs in Equation (1) is removed in the $E$-step. The removal of softmax in EM-DD greatly decreases the computation time. In addition, we believe that EM-DD helps avoid getting caught in local minimum since it makes major changes in the hypothesis when it switches which point is selected from a bag.

We now provide a sketch of the proof of convergence of EM-DD. Note that at each iteration $t$, given a set of instances selected in the E-step, the M-step will find a unique hypothesis ($h_t$) and corresponding DD ($dd_t$). At iteration $t + 1$, if $dd_{t+1} \leq dd_t$, the algorithm will terminate. Otherwise, $dd_{t+1} > dd_t$, which means that a different set of instances are selected. For the iteration to continue, the DD will decrease monotonically and the set of instances selected can not repeat. Since there are only finite number of sets to instances that can be selected at the E-step, the algorithm will terminate after a finite number of iterations.

However, there is no guarantee on the convergence rate of EM algorithms. We found that the NLDD($h, D$) usually decreases dramatically after the first several iterations and then begins to flatten out. From empirical tests we found that it is often beneficial to allow NLDD to increase slightly to escape a local minima and thus we used the less restrictive termination condition: $|dd_1 - dd_0| < 0.01 \cdot dd_0$ or the number of iterations is greater than 10. This modification reduces the training time while gaining comparable results. However, for this modification no convergence proof can be given without restricting the number of iterations.

## 4   Experimental results

In this section we summarize our experimental results. We begin by reporting our results for the two musk benchmark data sets provided by Dietterich et al. [4]. These data sets contain 166 feature vectors describing the surface for low-energy conformations of 92 molecules for Musk1 and 102 molecules for Musk2 where roughly half of the molecules are known to smell musky and the remainder are not. The Musk1 data set is smaller both in having fewer bags (i.e molecules) and many fewer instances per bag (an average of 6.0 for Musk1 versus 64.7 for Musk2). Prior to this work, the highly-tuned *iterated-discrim* algorithm of Dietterich et al. still gave the best performance on both Musk1 and Musk2. Maron and Lozano-Pérez [6]

```
Main(k, D)
    partition D = {D₁, D₂, ..., D₁₀};  //10-fold cross validation
    for (i = 1; i ≤ 10; i++)
        Dₜ = D - Dᵢ;       //Dₜ training data, Dᵢ validation data
        pick k random positive bags B₁, ..., Bₖ from Dₜ;
        let H₀ be the union of all instances from selected bags;
        for every instance Iⱼ ∈ H₀
            hⱼ = EM-DD(Iⱼ, Dₜ);
    eᵢ = min₀≤ⱼ≤‖H₀‖ {error(hⱼ, Dᵢ)};
    return avg(e₁, e₂, ..., e₁₀);

EM-DD(I, Dₜ)
    Let h = {h₁, ..., hₙ, s₁, ..., sₙ};          //initial hypothesis
    For each dimension d = 1, ..., n
        h_d = I_d;      s_d = 0.1;
    nldd₀ = +∞;       nldd₁ = NLDD(h, Dₜ);
    while (nldd₁ < nldd₀)
        for each bag Bᵢ ∈ Dₜ                  //E-step
            p*ᵢ = arg max_{Bᵢⱼ∈Bᵢ} Pr(Bᵢⱼ ∈ h);
            h' = arg max_{h∈H} ∏ᵢ Pr(ℓᵢ | h, p*ᵢ);        //M-step
            nldd₀ = nldd₁;       nldd₁ = NLDD(h', Dₜ);       h = h';
    return h;
```

Figure 1: Pseudo-code for EM-DD where $k$ indicates the number of different starting bags used, $\Pr(B_{ij} \in h) = \exp[-\sum_{d=1}^{n}(s_d(B_{ijd} - h_d))^2]$. $\Pr(\ell_i \mid h, p_i^*)$ is calculate as either $1 - |\ell_i - \Pr(p_i^* \in h)|$ (linear model) or $\exp[-(\ell_i - \Pr(p_i^* \in h))^2]$ (Gaussian-like model), where $\Pr(p_i^* \in h) = \max_{B_{ij} \in B_i} \Pr(B_{ij} \in h)$.

summarize the generally held belief that "The performance reported for iterated-discrim APR involves choosing parameters to maximize the *test set* performance and so probably represents an upper bound for accuracy on this (Musk1) data set."

EM-DD without tuning outperforms all previous algorithms. To be consistent with the way in which past results have been reported for the musk benchmarks we report the average accuracy of 10-fold cross-validation (which is the value returned by Main in Figure 1. EM-DD obtains an average accuracy of **96.8%** on Musk1 and **96.0%** on Musk2. A summary of the performance of different algorithms on the Musk1 and Musk2 data sets is given in Table 1. In addition, for both data sets, there are no false negative errors using EM-DD, which is important for the drug discovery application since the final hypothesis would be used to filter potential drugs and a false negative error means that a potential good drug molecule would not be tested and thus it is good to minimize such errors. As compared to the standard DD algorithm, EM-DD only used three random bags for Musk1 and two random bags for Musk2 (versus all positive bags used in DD) as the starting point of the algorithm. Also, unlike the results reported in [6] in which the threshold is tuned based on leave-one-out cross validation, for our reported results the threshold value (of 0.5) is not tuned. More importantly, EM-DD runs over 10 times faster than DD on Musk1 and over 100 times faster when applied to Musk2.

Table 1: Comparison of performance on Musk1 and Musk2 data sets as measured by giving the average accuracy across 10 runs using 10-fold cross validation.

| Algorithm | Musk1 accuracy | Musk 2 accuracy |
|---|---|---|
| **EM-DD** | **96.8%** | **96.0%** |
| Iterated-discrim [4] | 92.4% | 89.2% |
| Citation-kNN [11] | 92.4% | 86.3% |
| Bayesian-kNN [11] | 90.2% | 82.4% |
| Diverse density [6] | 88.9% | 82.5% |
| Multi-instance neural network [9] | 88.0% | 82.0% |
| Multinst [2] | 76.7% | 84.0% |

In addition to its superior performance on the musk data sets, EM-DD can handle real-value labeled data and produces real-value predictions. We present results using one real data set (*Affinity*) [1] that has real-value labels and several artificial data sets generated using the technique of our earlier work [1]. For these data sets, we used as our starting points the points from the bag with the highest DD value. The result are shown in Table 2. The Affinity data set has 283 features and 139 bags with an average of 32.5 points per bag. Only 29 bags have labels that were high enough to be considered as "positive." Using the Gaussian-like version of our generative model we obtained a squared loss of 0.0185 and with the linear model we performed slightly better with a loss of 0.0164. In contrast using the standard diverse density algorithm the loss was 0.0421. EM-DD also gained much better performance than DD on two artificial data (160.166.1a-S and 80.166.1a-S) where both algorithms were used[2]. The best result on Affinity data was obtained using a version of citation-$k$NN [1] that works with real-value data with the loss as 0.0124. We think that the affinity data set is well-suited for a nearest neighbor approach in that all of the negative bags have labels between 0.34 and 0.42 and so the actual predictions for the negative bags are better with citation-$k$NN.

To study the sensitivity of EM-DD to the number of relevant attributes and the size of the bags, tests were performed on artificial data sets with different number of relevant features and bag sizes. As shown in Table 2, similar to the DD algorithm [1], the performance of EM-DD degrades as the number of relevant features decreases. This behavior is expected since all scale factors are initialized to the same value and when most of the features are relevant less adjustment is needed and hence the algorithm is more likely to succeed. In comparison to DD, EM-DD is more robust against the change of the number of relevant features. For example, as shown in Figure 2, when the number of relevant features is 160 out of 166, both EM-DD and DD algorithms perform well with good correlation between the actual labels and predicted labels. However, when the number of relevant features decreases to 80, almost no correlation between the actual and predicted labels is found using DD, while EM-DD can still provide good predictions on the labels.

Intuitively, as the size of bags increases, more ambiguity is introduced to the data and the performance of algorithms is expected to go down. However, somewhat

Table 2: Performance on data with real-value labels measured as squared loss.

| Data set | # rel. features | #pts per bag | EM-DD | DD [1] |
|----------|-----------------|--------------|-------|--------|
| Affinity | – | 32.5 | .0164 | .0421 |
| 160.166.1a-S | 160 | 4 | .0014 | .0052 |
| 160.166.1b-S | 160 | 15 | .0013 | – |
| 160.166.1c-S | 160 | 25 | .0012 | – |
| 80.166.1a-S | 80 | 4 | .0029 | .1116 |
| 80.166.1b-S | 80 | 15 | .0023 | – |
| 80.166.1c-S | 80 | 25 | .0022 | – |
| 40.166.1a-S | 40 | 4 | .0038 | – |
| 40.166.1b-S | 40 | 15 | .0026 | – |
| 40.166.1c-S | 40 | 25 | .0037 | – |

surprisingly, the performance of EM-DD actually improves as the number of examples per bag increases. We believe that this is partly due to the fact that with few points per bag the chance that a bad starting point has the highest diverse density is much higher than when the bags are large. In addition, in contrast to the standard diverse density algorithm, the overall time complexity of EM-DD does not go up as the size of the bags increased, since after the instance selection (E-step), the time complexities of the dominant M-step are essentially the same for data sets with different bag sizes. The fact that EM-DD scales up well to large bag sizes in both performance and running time is very important for real drug-discovery applications in which the bags can be quite large.

## 5 Future directions

There are many avenues for future work. We believe that EM-DD can be refined to obtain better performance by finding alternate ways to select the initial hypothesis and scale factors. One option would be to use the result from a different learning algorithm as the starting point then use EM-DD to refine the hypothesis. We are currently studying the application of the EM-DD algorithm to other domains such as content-based image retrieval. Since our algorithm is based on the diverse density likelihood measurement we believe that it will perform well on all applications in which the standard diverse density algorithm has worked well. In addition, EM-DD and MI regression [10] presented a framework to convert the multiple-instance data to single-instance data, where supervised learning algorithms can be applied. We are currently working on using this general methodology to develop new MI learning techniques based on supervised learning algorithms and EM.

### Acknowledgments
The authors gratefully acknowledge the support NSF grant CCR-9988314. We thank Dan Dooly for many useful discussions. We also thank Jonathan Greene who provided us with the Affinity data set.

## Footnotes

[1]Jonathan Greene from CombiChem provided us with the Affinity data set. However, due to the proprietary nature of it we cannot make it publicly available.

[2]See Amar et al. [1] for a description of these two data sets.

### References

[1] Amar, R.A., Dooly, D.R., Goldman, S.A. & Zhang, Q. (2001). Multiple-Instance Learning of Real-Valued Data. *Proceedings 18th International Conference on Machine Learning*, pp. 3–10. San Francisco, CA: Morgan Kaufmann.

[2] Auer, P. (1997) On learning from mult-instance examples: Empirical evaluation of a theoretical approach. *Proceedings 14th International Conference on Machine Learning*,

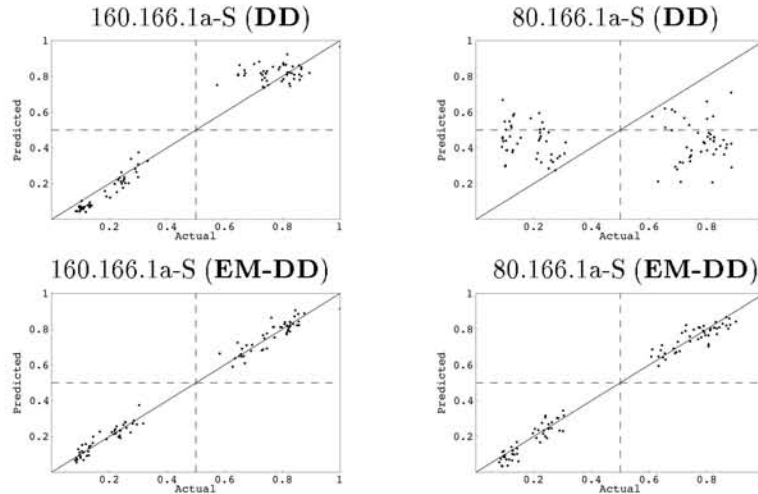

Figure 2: Comparison of EM-DD and DD on real-value labeled artificial data with different number of relevant features. The $x$-axis corresponds to the *actual* label and $y$-axis gives the *predicted* label.

pp. 21-29. San Francisco, CA: Morgan Kaufmann.

[3] Dempster, A.P., Laird, N.M., & Rubin, D.B. (1977). Maximum likelihood from incomplete data via the EM algorithm. *Journal of the Royal Statistics Society*, Series B, **39**(1): 1-38.

[4] Dietterich, T. G., Lathrop, R. H., & Lozano-Pérez, T. (1997). Solving the multiple-instance problem with axis-parallel rectangles. *Artificial Intelligence*, **89**(1-2): 31-71.

[5] Maron, O. (1998). *Learning from Ambiguity.* Doctoral dissertation, MIT, AI Technical Report 1639.

[6] Maron, O. & Lozano-Pérez, T. (1998). A framework for multiple-instance learning. *Neural Information Processing Systems 10.* Cambridge, MA: MIT Press.

[7] Maron, O. & Ratan, A. (1998). Multiple-instance learning for natural scene classification. *Proceedings 15th International Conference on Machine Learning*, pp. 341-349. San Francisco, CA: Morgan Kaufmann.

[8] Press, W.H., Teukolsky, S.A., Vetterling, W.T., and Flannery, B.P. (1992). *Numerical Recipes in C: the art of scientific computing.* Cambridge University Press, New York, second edition.

[9] Ramon, J. & L. De Raedt. (2000). Multi instance neural networks. *Proceedings of ICML-2000 workshop on "Attribute-Value and Relational Learning.*

[10] Ray, S. & Page, D. (2001). Multiple-Instance Regression. *Proceedings 18th International Conference on Machine Learning*, pp. 425–432. San Francisco, CA: Morgan Kaufmann.

[11] Ruffo, G. (2000). *Learning single and multiple instance decision trees for computer security applications.* Doctoral dissertation. Department of Computer Science, University of Turin, Torino, Italy.

[12] Wang, J. & Zucker, J.-D. (2000). Solving the Multiple-Instance Learning Problem: A Lazy Learning Approach. *Proceedings 17th International Conference on Machine Learning*, pp. 1119-1125. San Francisco, CA: Morgan Kaufmann.
